# Learning first-order Markov models for control

**Pieter Abbeel**
Computer Science Department
Stanford University
Stanford, CA 94305

**Andrew Y. Ng**
Computer Science Department
Stanford University
Stanford, CA 94305

## Abstract

First-order Markov models have been successfully applied to many problems, for example in modeling sequential data using Markov chains, and modeling control problems using the Markov decision processes (MDP) formalism. If a first-order Markov model's parameters are estimated from data, the standard maximum likelihood estimator considers only the first-order (single-step) transitions. But for many problems, the first-order conditional independence assumptions are not satisfied, and as a result the higher order transition probabilities may be poorly approximated. Motivated by the problem of learning an MDP's parameters for control, we propose an algorithm for learning a first-order Markov model that explicitly takes into account higher order interactions during training. Our algorithm uses an optimization criterion different from maximum likelihood, and allows us to learn models that capture longer range effects, but without giving up the benefits of using first-order Markov models. Our experimental results also show the new algorithm outperforming conventional maximum likelihood estimation in a number of control problems where the MDP's parameters are estimated from data.

## 1 Introduction

First-order Markov models have enjoyed numerous successes in many sequence modeling and in many control tasks, and are now a workhorse of machine learning.[1] Indeed, even in control problems in which the system is suspected to have hidden state and thus be non-Markov, a fully observed Markov decision process (MDP) model is often favored over partially observable Markov decision process (POMDP) models, since it is significantly easier to solve MDPs than POMDPs to obtain a controller. [5]

When the parameters of a Markov model are not known a priori, they are often estimated from data using maximum likelihood (ML) (and perhaps smoothing). However, in many applications the dynamics are not truly first-order Markov, and the ML criterion may lead to poor modeling performance. In particular, we will show that the ML model fitting criterion explicitly considers only the first-order (one-step) transitions. If the dynamics are truly governed by a first-order system, then the longer-range interactions would also be well modeled. But if the system is not first-order, then interactions on longer time scales are often poorly approximated by a model fit using maximum likelihood. In reinforcement learning and control tasks where the goal is to maximize our long-term expected rewards, the predictive accuracy of a model on long time scales can have a significant impact on the attained performance.

As a specific motivating example, consider a system whose dynamics are governed by a random walk on the integers. Letting $S_t$ denote the state at time $t$, we initialize the system to $S_0 = 0$, and let $S_t = S_{t-1} + \varepsilon_t$, where the increments $\varepsilon_t \in \{-1, +1\}$ are equally likely to be $-1$ or $+1$. Writing $S_t$ in terms of only the $\varepsilon_t$'s, we have $S_t = \varepsilon_1 + \cdots + \varepsilon_t$. Thus, if the increments are independent, we have $\mathrm{Var}(S_T) = T$. However if the increments are perfectly correlated (so $\varepsilon_1 = \varepsilon_2 = \cdots$ with probability 1), then $\mathrm{Var}(S_T) = T^2$. So, depending on the correlation between the increments, the expected value $\mathrm{E}[|S_T|]$ can be either $O(\sqrt{T})$ or $O(T)$. Further, regardless of the true correlation in the data, using maximum likelihood (ML) to estimate the model parameters from training data would return the same model with $\mathrm{E}[|S_T|] = O(\sqrt{T})$.

To see how these effects can lead to poor performance on a control task, consider learning to control a vehicle (such as a car or a helicopter) under disturbances $\varepsilon_t$ due to very strong winds. The influence of the disturbances on the vehicle's position over one time step may be small, but if the disturbances $\varepsilon_t$ are highly correlated, their cumulative effect over time can be substantial. If our model completely ignores these correlations, we may overestimate our ability to control the vehicle (thinking our variance in position is $O(T)$ rather than $O(T^2)$), and try to follow overly narrow/dangerous paths.

Our motivation also has parallels in the debate on using discriminative vs. generative algorithms for supervised learning. There, the consensus (assuming there is ample training data) seems to be that it is usually better to directly minimize the loss with respect to the ultimate performance measure, rather than an intermediate loss function such as the likelihood of the training data. (See, e.g., [16, 9].) This is because the model (no matter how complicated) is almost always not completely "correct" for the problem data. By analogy, when modeling a dynamical system for a control task, we are interested in having a model that accurately predicts the performance of different control policies—so that it can be used to select a good policy—and not in maximizing the likelihood of the observed sequence data.

In related work, robust control offers an alternative family of methods for accounting for model inaccuracies, specifically by finding controllers that work well for a large class of models. (E.g., [13, 17, 3].) Also, in applied control, some practitioners manually adjust their model's parameters (particularly the model's noise variance parameters) to obtain a model which captures the variability of the system's dynamics. Our work can be viewed as proposing an algorithm that gives a more structured approach to estimating the "right" variance parameters. The issue of time scales has also been addressed in hierarchical reinforcement learning (e.g., [2, 15, 11]), but most of this work has focused on speeding up exploration and planning rather than on accurately modeling non-Markovian dynamics.

The rest of this paper is organized as follows. We define our notation in Section 2, then formulate the model learning problem ignoring actions in Section 3, and propose a learning algorithm in Section 4. In Section 5, we extend our algorithm to incorporate actions. Section 6 presents experimental results, and Section 7 concludes.

## 2 Preliminaries

If $x \in \mathbb{R}^n$, then $x_i$ denotes the $i$-th element of $x$. Also, let $j{:}k = [j \; j{+}1 \; j{+}2 \; \cdots k{-}1 \; k]^T$. For any $k$-dimensional vector of indices $I \in \mathbf{N}^k$, we denote by $x_I$ the $k$-dimensional vector with the subset of $x$'s entries whose indices are in $I$. For example, if $x = [0.0 \; 0.1 \; 0.2 \; 0.3 \; 0.4 \; 0.5]^T$, then $x_{0:2} = [0.0 \; 0.1 \; 0.2]^T$.

A finite-state decision process (DP) is a tuple $(S, A, T, \gamma, D, R)$, where $S$ is a finite set of states; $A$ is a finite set of actions; $T = \{P(S_{t+1} = s' | S_{0:t} = s_{0:t}, A_{0:t} = a_{0:t})\}$ is a set of state transition probabilities (here, $P(S_{t+1} = s' | S_{0:t} = s_{0:t}, A_{0:t} = a_{0:t})$ is the probability of being in a state $s' \in S$ at time $t + 1$ after having taken actions $a_{0:t} \in A^{t+1}$ in states $s_{0:t} \in S^{t+1}$ at times $0 : t$); $\gamma \in [0, 1)$ is a discount factor; $D$ is the initial state distribution, from which the initial state $s_0$ is drawn; and $R : S \mapsto \mathbb{R}$ is the reward function. We assume all rewards are bounded in absolute value by $R_{\max}$. A DP is not necessarily Markov.

A policy $\pi$ is a mapping from states to probability distributions over actions. Let $V^\pi(s) = \mathrm{E}[\sum_{t=0}^{\infty} \gamma^t R(s_t)|\pi, s_0 = s]$ be the usual value function for $\pi$. Then the utility of $\pi$ is

$$U(\pi) = \mathrm{E}_{s_0 \sim D}[V^\pi(s_0)] = \mathrm{E}[\textstyle\sum_{t=0}^{\infty} \gamma^t R(s_t)|\pi] = \sum_{t=0}^{\infty} \gamma^t \sum_{s_t} P(S_t = s_t|\pi)R(s_t).$$

The second expectation above is with respect to the random state sequence $s_0, s_1, \ldots$ drawn by starting from $s_0 \sim D$, picking actions according to $\pi$ and transitioning according to $P$.

Throughout this paper, $P_{\hat\theta}$ will denote some estimate of the transition probabilities. We denote by $\hat U(\pi)$ the utility of the policy $\pi$ in an MDP whose first-order transition probabilities are given by $P_{\hat\theta}$ (and similarly $\hat V^\pi$ the value function in the same MDP). Thus, we have[2]

$$\hat U(\pi) = \hat{\mathrm{E}}_{s_0 \sim D}[\hat V^\pi(s_0)] = \hat{\mathrm{E}}[\textstyle\sum_{t=0}^{\infty} \gamma^t R(s_t)|\pi] = \sum_{t=0}^{\infty} \gamma^t \sum_{s_t} P_{\hat\theta}(S_t = s_t|\pi)R(s_t).$$

Note that if $|U(\pi) - \hat U(\pi)| \le \varepsilon$ for all $\pi$, then finding the optimal policy in the estimated MDP that uses parameters $P_{\hat\theta}$ (using value iteration or any other algorithm) will give a policy whose utility is within $2\varepsilon$ of the optimal utility. [6]

For stochastic processes without decisions/actions, we will use the same notation but drop the conditioning on $\pi$. Often we will also abbreviate $P(S_t = s_t)$ by $P(s_t)$.

## 3   Problem Formulation

To simplify our exposition, we will begin by considering stochastic processes that do not have decisions/actions. Section 5 will discuss how actions can be incorporated into the model.

We first consider how well $\hat V(s_0)$ approximates $V(s_0)$. We have

$$
\begin{aligned}
|\hat V(s_0) - V(s_0)| &= \left| \sum_{t=0}^{\infty} \gamma^t \sum_{s_t} P_{\hat\theta}(s_t|s_0)R(s_t) - \sum_{t=0}^{\infty} \gamma^t \sum_{s_t} P(s_t|s_0)R(s_t) \right| \\
&\le R_{\max} \sum_{t=0}^{\infty} \gamma^t \sum_{s_t} \left| P_{\hat\theta}(s_t|s_0) - P(s_t|s_0) \right|.
\end{aligned}
\tag{1}
$$

So, to ensure that $\hat V(s_0)$ is an accurate estimate of $V(s_0)$, we would like the parameters $\hat\theta$ of the model to minimize the right hand side of (1). The term $\sum_{s_t} |P_{\hat\theta}(s_t|s_0) - P(s_t|s_0)|$ is exactly (twice) the variational distance between the two conditional distributions $P_{\hat\theta}(\cdot|s_0)$ and $P(\cdot|s_0)$. Unfortunately $P$ is not known when learning from data. We only get to observe state sequences sampled according to $P$. This makes Eqn. (1) a difficult criterion to optimize. However, it is well known that the variational distance is upper bounded by a function of the KL-divergence. (See, e.g., [1].) The KL-divergence between $P$ and $P_{\hat\theta}$ can be estimated (up to a constant) as the log-likelihood of a sample. So, given a training sequence $s_{0:T}$ sampled from $P$, we propose to estimate the transition probabilities $P_{\hat\theta}$ by

$$\hat\theta = \arg\max_\theta \sum_{t=0}^{T-1} \sum_{k=1}^{T-t} \gamma^k \log P_\theta(s_{t+k}|s_t). \tag{2}$$

Note the difference between this and the standard maximum likelihood (ML) estimate. Since we are using a model that is parameterized as a first-order Markov model, the probability of the data under the model is given by $P_\theta(s_0, \ldots, s_T) = P_\theta(s_T|s_{T-1})P_\theta(s_{T-1}|s_{T-2}) \ldots P_\theta(s_1|s_0)D(s_0)$ (where $D$ is the initial state distribution). By definition, maximum likelihood (ML) chooses the parameters $\theta$ that maximize the probability of the observed data. Taking logs of the probability above, (and ignoring $D(s_0)$, which is usually parameterized separately), we find that the ML estimate is given by

$$\hat\theta = \arg\max_\theta \sum_{t=0}^{T-1} \log P_\theta(s_{t+1}|s_t). \tag{3}$$

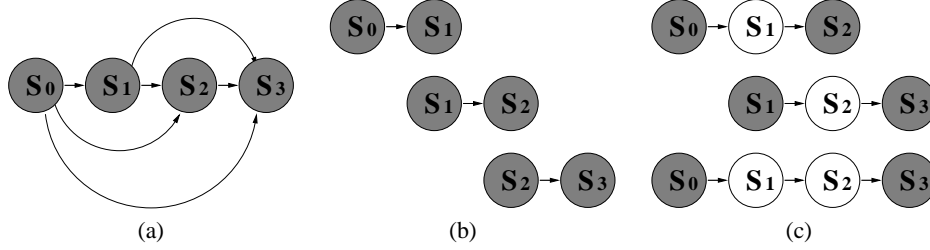

Figure 1: (a) A length four training sequence. (b) ML estimation for a first-order Markov model optimizes the likelihood of the second node given the first node in each of the length two subsequences. (c) Our objective (Eqn. 2) also includes the likelihood of the last node given the first node in each of these three longer subsequences of the data. (White nodes represent unobserved variables, shaded nodes represent observed variables.)

All the terms above are of the form $P_\theta(s_{t+1}|s_t)$. Thus, the ML estimator explicitly considers, and tries to model well, *only the observed one-step transitions*. In Figure 1 we use Bayesian network notation to illustrate the difference between the two objectives for a training sequence of length four. Figure 1(a) shows the training sequence, which can have arbitrary dependencies. Maximum likelihood (ML) estimation maximizes $f_{ML}(\theta) = \log P_\theta(s_1|s_0) + \log P_\theta(s_2|s_1) + \log P_\theta(s_3|s_2)$. Figure 1(b) illustrates the interactions modeled by ML. Ignoring $\gamma$ for now, for this example our objective (Eqn. 2) is $f_{ML}(\theta) + \log P_\theta(s_2|s_0) + \log P_\theta(s_3|s_1) + \log P_\theta(s_3|s_0)$. Thus, it takes into account both the interactions in Figure 1(b) as well as the longer-range ones in Figure 1(c).

## 4   Algorithm

We now present an EM algorithm for optimizing the objective in Eqn. (2) for a first-order Markov model.[3] Our algorithm is derived using the method of [7]. (See the Appendix for details.) The algorithm iterates between the following two steps:

- *E-step: Compute expected counts*
  - $\forall i, j \in S$, set $stats(j,i) = 0$
  - $\forall t : 0 \le t \le T - 1, \forall k : 1 \le k \le T - t, \forall l : 0 \le l \le k - 1, \forall i, j \in S$
    $stats(j,i) \mathrel{+}= \gamma^k P_{\hat\theta}(S_{t+l+1} = j, S_{t+l} = i | S_t = s_t, S_{t+k} = s_{t+k})$
- *M-step: Re-estimate model parameters*

  Update $\hat\theta$ such that $\forall i, j \in S$, $P_{\hat\theta}(j|i) = stats(j,i)/\sum_{k \in S} stats(k,i)$

Prior to starting EM, the transition probabilities $P_{\hat\theta}$ can be initialized with the first-order transition counts (i.e., the ML estimate of the parameters), possibly with smoothing.[4]

Let us now consider more carefully the computation done in the E-step for one specific pair of values for $t$ and $k$ (corresponding to one term $\log P_\theta(s_{t+k}|s_t)$ in Eqn. 2). For $k \ge 2$, as in the forward-backward algorithm for HMMs (see, e.g., [12, 10]), the pairwise marginals can be computed by a forward propagation (computing the forward messages), a backward propagation (computing the backward messages), and then combining the forward and backward messages.[5] Forward and backward messages are computed recursively:

$$\text{for} \quad l = 1 \text{ to } k - 1, \quad \forall i \in S \quad m_{\rightarrow t+l}(i) = \sum_{j \in S} m_{\rightarrow t+l-1}(j) P_{\hat\theta}(i|j), \qquad (4)$$

$$\text{for} \quad l = k - 1 \text{ down to } 1, \quad \forall i \in S \quad m_{t+l \leftarrow}(i) = \sum_{j \in S} m_{t+l+1 \leftarrow}(j) P_{\hat\theta}(j|i), \qquad (5)$$

where we initialize $m_{\to t}(i) = \mathbf{1}\{i = s_t\}$, and $m_{t+k\leftarrow}(i) = \mathbf{1}\{i = s_{t+k}\}$. The pairwise marginals can be computed by combining the forward and backward messages:

$$P_{\hat{\theta}}(S_{t+l+1} = j, S_{t+l} = i | S_t = s_t, S_{t+k} = s_{t+k}) = m_{\to t+l}(i)P_{\hat{\theta}}(j|i)m_{t+l+1\leftarrow}(j). \quad (6)$$

For the term $\log P_\theta(s_{t+k}|s_t)$, we end up performing $2(k-1)$ message computations, and combining messages into pairwise marginals $k-1$ times. Doing this for all terms in the objective results in $O(T^3)$ message computations and $O(T^3)$ computations of pairwise marginals from these messages. In practice, the objective (2) can be approximated by considering only the terms in the summation with $k \leq H$, where $H$ is some time horizon.[6] In this case, the computational complexity is reduced to $O(TH^2)$.

## 4.1 Computational Savings

The following observation leads to substantial savings in the number of message computations. The forward messages computed for the term $\log P_\theta(s_{t+k}|s_t)$ depend only on the value of $s_t$. So the forward messages computed for the terms $\{\log P_\theta(s_{t+k}|s_t)\}_{k=1}^H$ are the same as the forward messages computed just for the term $\log P_\theta(s_{t+H}|s_t)$. A similar observation holds for the backward messages. As a result, we need to compute only $O(TH)$ messages (as opposed to $O(TH^2)$ in the naive algorithm).

The following observation leads to further, (even more substantial) savings. Consider two terms in the objective $\log P_\theta(s_{t_1+k}|s_{t_1})$ and $\log P_\theta(s_{t_2+k}|s_{t_2})$. If $s_{t_1} = s_{t_2}$ and $s_{t_1+k} = s_{t_2+k}$, then both terms will have exactly the same pairwise marginals and contribution to the expected counts. So expected counts have to be computed only once for every triple $i, j, k$ for which $(S_t = i, S_{t+k} = j)$ occurs in the training data. As a consequence, the running time for each iteration (once we have made an initial pass over the data to count the number of occurrences of the triples) is only $O(|S|^2 H^2)$, which is independent of the size of the training data.

## 5 Incorporating actions

In decision processes, actions influence the state transition probabilities. To generate training data, suppose we choose an exploration policy and take actions in the DP using this policy. Given the resulting training data, and generalizing Eqn. (2) to incorporate actions, our estimator now becomes

$$\hat{\theta} = \arg\max_\theta \sum_{t=0}^{T-1} \sum_{k=1}^{T-t} \gamma^k \log P_\theta(s_{t+k}|s_t, a_{t:t+k-1}). \quad (7)$$

The EM algorithm is straightforwardly extended to this setting, by conditioning on the actions during the E-step, and updating state-action transition probabilities $P_\theta(j|i, a)$ in the M-step.

As before, forward messages need to be computed only once for each value of $t$, and backward messages only once for each value of $t + k$. However achieving the more substantial savings, as described in the second paragraph of Section 4.1, is now more difficult. In particular, now the contribution of a triple $i, j, k$ (one for which $(S_t = i, S_{t+k} = j)$ occurs in the training data) depends on the action sequence $a_{t:t+k-1}$. The number of possible sequences of actions $a_{t:t+k-1}$ grows exponentially with $k$.

If, however, we use a deterministic exploration policy to generate the training data (more specifically, one in which the action taken is a deterministic function of the current state), then we can again obtain these computational advantages: Counts of the number of occurrences of the triples described previously are now again a sufficient statistic. However, a single deterministic exploration policy, by definition, cannot explore all state-action pairs. Thus, we will instead use a combination of several deterministic exploration policies, which jointly can explore all state-action pairs. In this case, the running time for the E-step becomes $O(|S|^2 H^2 |\Pi|)$, where $|\Pi|$ is the number of different deterministic exploration policies used. (See Section 6.2 for an example.)

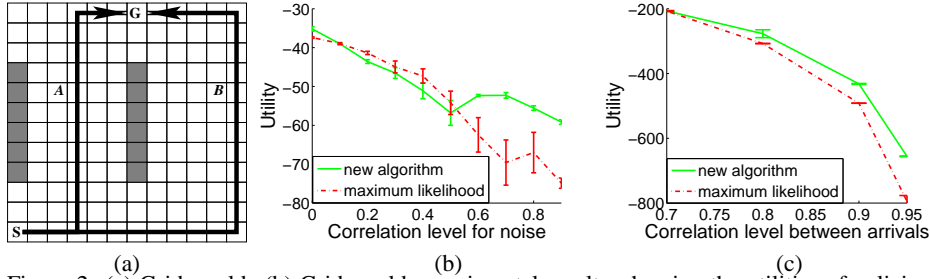

<div align="center">(a)          (b)          (c)</div>

Figure 2: (a) Grid-world. (b) Grid-world experimental results, showing the utilities of policies obtained from the MDP estimated using ML (dash-dot line), and utilities of policies obtained from the MDP estimated using our objective (solid line). Results shown are means over 5 independent trials, and the error bars show one standard error for the mean. The horizontal axis (correlation level for noise) corresponds to the parameter $q$ in the experiment description. (c) Queue experiment, showing utilities obtained using ML (dash-dot line), and using our algorithm (solid line). Results shown are means over 5 independent trials, and the error bars show one standard error for the mean. The horizontal axis (correlation level between arrivals) corresponds to the parameter $b$ in the experiment description. (Shown in color, where available.)

# 6 Experiments

In this section, we empirically study the performance of model fitting using our proposed algorithm, and compare it to the performance of ordinary ML estimation.

## 6.1 Shortest vs. safest path

Consider an agent acting for 100 time steps in the grid-world in Figure 2(a). The initial state is marked by S, and the absorbing goal state by G. The reward is -500 for the gray squares, and -1 elsewhere. This DP has four actions that (try to) move in each of the four compass directions, and succeed with probability $1 - p$. If an action is not successful, then the agent's position transitions to one of the neighboring squares. Similar to our example in Section 1, the random transitions (resulting from unsuccessful actions) may be correlated over time. In this problem, if there is no noise ($p = 0$), the optimal policy is to follow one of the shortest paths to the goal that do not pass through gray squares, such as path $A$. For higher noise levels, the optimal policy is to stay as far away as possible from the gray squares, and try to follow a longer path such as $B$ to the goal.[7] At intermediate noise levels, the optimal policy is strongly dependent on how correlated the noise is between successive time steps. The larger the correlation, the more dangerous path $A$ becomes (for reasons similar to the random walk example in Section 1). In our experiments, we compare the behavior of our algorithm and ML estimation with different levels of noise correlation.[8]

Figure 2(b) shows the utilities obtained by the two different models, under different degrees of correlation in the noise. The two algorithms perform comparably when the correlation is weak, but our method outperforms ML when there is strong correlation. Empirically, when the noise correlation is high, our algorithm seems to be fitting a first-order model with a larger "effective" noise level. When the resulting estimated MDP is solved, this gives more cautious policies, such as ones more inclined to choose path $B$ over $A$. In contrast, the ML estimate performs poorly in this problem because it tends to underestimate how far sideways the agent tends to move due to the noise (cf. the example in Section 1).

## 6.2 Queue

We consider a service queue in which the average arrival rate is $p$. Thus, $p = P(\text{a customer arrives in one time step})$. Also, for each action $i$, let $q_i$ denote the service rate under that action (thus, $q_i = P(\text{a customer is served in one time step}|\text{action} = i)$). In our problem, there are three service rates $q_0 < q_1 < q_2$ with respective rewards $0, -1, -10$. The maximum queue size is 20, and the reward for any state of the queue is 0, except when the queue becomes full, which results in a reward of -1000. The service rates are $q_0 = 0$, $q_1 = p$ and $q_2 = 0.75$. So the inexpensive service rate $q_1$ is sufficient to keep up with arrivals on average. However, even though the average arrival rate is $p$, the arrivals come in "bursts," and even the high service rate $q_2$ is insufficient to keep the queue small during the bursts of many consecutive arrivals.[9]

Experimental results on the queue are shown in Figure 2(c). We plot the utilities obtained using each of the two algorithms for high arrival correlations. (Both algorithms perform essentially identically at lower correlation levels.) We see that the policies obtained with our algorithm consistently outperform those obtained using maximum likelihood to fit the model parameters. As expected, the difference is more pronounced for higher correlation levels, i.e., when the true model is less well approximated by a first-order model.

For learning the model parameters, we used three deterministic exploration policies, each corresponding to always taking one of the three actions. Thus, we could use the more efficient version of the algorithm described in the second paragraph of Section 4.1 and at the end of Section 5. A single EM iteration for the experiments on the queue took 6 minutes for the original version of the algorithm, but took only 3 seconds for the more efficient version; this represents more than a 100-fold speedup.

## 7 Conclusions

We proposed a method for learning a first-order Markov model that captures the system's dynamics on longer time scales than a single time step. In our experiments, this method was also shown to outperform the standard maximum likelihood model. In other experiments, we have also successfully applied these ideas to modeling the dynamics of an autonomous RC car. (Details will be presented in a forthcoming paper.)

## Footnotes

[1] To simplify the exposition, in this paper we will consider only first-order Markov models. However, the problems we describe in this paper also arise with higher order models and with more structured models (such as dynamic Bayesian networks [4, 10] and mixed memory Markov models [8, 14]), and it is straightforward to extend our methods and algorithms to these models.

[2]Since $P_{\hat\theta}$ is a first-order model, it explicitly parameterizes only $P_{\hat\theta}(S_{t+1} = s_{t+1}|S_t = s_t, A_t = a_t)$. We use $P_{\hat\theta}(S_t = s_t|\pi)$ to denote the probability that $S_t = s_t$ in an MDP with one-step transition probabilities $P_{\hat\theta}(S_{t+1} = s_{t+1}|S_t = s_t, A_t = a_t)$ and initial state distribution $D$ when acting according to the policy $\pi$.

[3]Using higher order Markov models or more structured models (such as dynamic Bayesian networks [4, 10] or mixed memory Markov models [8, 14]) offer no special difficulties, though the notation becomes more involved and the inference (in the E-step) might become more expensive.

[4]A parameter $P_{\hat\theta}(j|i)$ initialized to zero will remain zero throughout successive iterations of EM. If this is undesirable, then smoothing could be used to eliminate zero initial values.

[5]Note that the special case $k = 1$ (and thus $l = 0$) does not require inference. In this case we simply have $P_{\hat\theta}(S_{t+1} = j, S_t = i | S_t = s_t, S_{t+1} = s_{t+1}) = \mathbf{1}\{i = s_t\}\mathbf{1}\{j = s_{t+1}\}$.

[6]Because of the discount term $\gamma^k$ in the objective (2), one can safely truncate the summation over $k$ after about $O(1/(1-\gamma))$ terms without incurring too much error.

[7]For very high noise levels (e.g. $p = 0.99$) the optimal policy is qualitatively different again.

[8]Experimental details: The noise is governed by an (unobserved) Markov chain with four states corresponding to the four compass directions. If an action at time $t$ is not successful, the agent moves in the direction corresponding to the state of this Markov chain. On each step, the Markov chain stays in the current state with probability $q$, and transitions with probability $1 - q$ uniformly to any of the four states. Our experiments are carried out varying $q$ from 0 (low noise correlation) to 0.9 (strong noise correlation). A 200,000 length state-action sequence for the grid-world, generated using a random exploration policy, was used for model fitting, and a constant noise level $p = 0.3$ was used in the experiments. Given a learned MDP model, value iteration was used to find the optimal policy for it. To reduce computation, we only included the terms of the objective (Eqn. 7) for which $k = 10$.

## References

[1] T. M. Cover and J. A. Thomas. *Elements of Information Theory*. Wiley, 1991.

[2] T. G. Dietterich. Hierarchical reinforcement learning with the MAXQ value function decomposition. *JAIR*, 2000.

[3] P. Gahinet, A. Nemirovski, A. Laub, and M. Chilali. *LMI Control Toolbox*. Natick, MA, 1995.

[4] Z. Ghahramani. Learning dynamic Bayesian networks. In *Adaptive Processing of Sequences and Data Structures*, pages 168–197. Springer-Verlag, 1998.

---

[9]Experimental details: The true process has two different (hidden) modes for arrivals. The first mode has a very low arrival rate, and the second mode has a very high arrival rate. We denote the steady state distribution over the two modes by $(\phi_1, \phi_2)$. (I.e., the system spends a fraction $\phi_1$ of the time in the low arrival rate mode, and a fraction $\phi_2 = 1 - \phi_1$ of the time in high arrival rate mode.) Given the steady state distribution, the state transition matrix $[a \; 1 - a; 1 - b \; b]$ has only one remaining degree of freedom, which (essentially) controls how often the system switches between the two modes. (Here, $a$ [resp. $b$] is the probability, if we are in the slow [resp. fast] mode, of staying in the same mode the next time step.) More specifically, assuming $\phi_1 > \phi_2$, we have $b \in [0, 1]$, $a = 1 - (1 - b)\phi_2/\phi_1$. The larger $b$ is, the more slowly the system switches between modes. Our experiments used $\phi_1 = 0.8, \phi_2 = 0.2, P(\text{arrival}|\text{mode 1}) = 0.01, P(\text{arrival}|\text{mode 2}) = 0.99$. This means $b = 0.2$ gives independent arrival modes for consecutive time steps. In our experiments, $q_0 = 0$, and $q_1$ was equal to the average arrival rate $p = \phi_1 P(\text{arrival}|\text{mode 1}) + \phi_2 P(\text{arrival}|\text{mode 2})$. Note that the highest service rate $q_2(= 0.75)$ is lower than the fast mode's arrival rate. Training data was generated using 8000 simulations of 25 time steps each, in which the queue length is initialized randomly, and the same (randomly chosen) action is taken on all 25 time steps. To reduce computational requirements, we only included the terms of the objective (Eqn. 7) for which $k = 20$. We used a discount factor $\gamma = .95$ and approximated utilities by truncating at a finite horizon of 100. Note that although we explain the queuing process by arrival/departure rates, the algorithm learns full transition matrices for each action, and not only the arrival/departure rates.

[5] L. P. Kaelbling, M. L. Littman, and A. R. Cassandra. Planning and acting in partially observable stochastic domains. *Artificial Intelligence*, 101, 1998.

[6] M. Kearns, Y. Mansour, and A. Y. Ng. Approximate planning in large POMDPs via reusable trajectories. In *NIPS 12*, 1999.

[7] R. Neal and G. Hinton. A view of the EM algorithm that justifies incremental, sparse, and other variants. In *Learning in Graphical Models*, pages 355–368. MIT Press, 1999.

[8] H. Ney, U. Essen, and R. Kneser. On structuring probabilistic dependencies in stochastic language modeling. *Computer Speech and Language*, 8, 1994.

[9] A. Y. Ng and M. I. Jordan. On discriminative vs. generative classifiers: A comparison of logistic regression and naive Bayes. In *NIPS 14*, 2002.

[10] J. Pearl. *Probabilistic Reasoning in Intelligent Systems: Networks of Plausible Inference*. Morgan Kauffman, 1988.

[11] D. Precup, R. S. Sutton, and S. Singh. Theoretical results on reinforcement learning with temporally abstract options. In *Proc. ECML*, 1998.

[12] L. R. Rabiner. A tutorial on hidden Markov models and selected applications in speech recognition. *Proceedings of the IEEE*, 77, 1989.

[13] J. K. Satia and R. L. Lave. Markov decision processes with uncertain transition probabilities. *Operations Research*, 1973.

[14] L. K. Saul and M. I. Jordan. Mixed memory Markov models: decomposing complex stochastic processes as mixtures of simpler ones. *Machine Learning*, 37, 1999.

[15] R. S. Sutton. TD models: Modeling the world at a mixture of time scales. In *Proc. ICML*, 1995.

[16] V. N. Vapnik. *Statistical Learning Theory*. John Wiley & Sons, 1998.

[17] C. C. White and H. K. Eldeib. Markov decision processes with imprecise transition probabilities. *Operations Research*, 1994.

## Appendix: Derivation of EM algorithm

This Appendix derives the EM algorithm that optimizes Eqn. (7). The derivation is based on [7]'s method. Note that because of discounting, the objective is slightly different from the standard setting of learning the parameters of a Markov chain with unobserved variables in the training data.

Since we are using a first-order model, we have $P_{\hat{\theta}}(s_{t+k}|s_t, a_{t:t+k-1}) = \sum_{S_{t+1:t+k-1}} P_{\hat{\theta}}(s_{t+k}|S_{t+k-1}, a_{t+k-1}) P_{\hat{\theta}}(S_{t+k-1}|S_{t+k-2}, \; a_{t+k-2}) \ldots P_{\hat{\theta}}(S_{t+1}|s_t, a_t)$. Here, the summation is over all possible state sequences $S_{t+1:t+k-1}$. So we have

$$\sum_{t=0}^{T-1} \sum_{k=1}^{T-t} \gamma^k \log P_{\hat{\theta}}(s_{t+k}|s_t, a_{t:t+k-1})$$

$$= \sum_{t=0}^{T-1} \gamma \log P_{\hat{\theta}}(s_{t+1}|s_t, a_t) + \sum_{t=0}^{T-1} \sum_{k=2}^{T-t} \gamma^k \log \sum_{S_{t+1:t+k-1}} \frac{Q_{t,k}(S_{t+1:t+k-1})}{Q_{t,k}(S_{t+1:t+k-1})}$$
$$P_{\hat{\theta}}(s_{t+k}|S_{t+k-1}, a_{t+k-1}) P_{\hat{\theta}}(S_{t+k-1}|S_{t+k-2}, a_{t+k-2}) \ldots P_{\hat{\theta}}(S_{t+1}|s_t, a_t)$$

$$\geq \sum_{t=0}^{T-1} \gamma \log P_{\hat{\theta}}(s_{t+1}|s_t, a_t) + \sum_{t=0}^{T-1} \sum_{k=2}^{T-t} \gamma^k Q_{t,k}(S_{t+1:t+k-1})$$
$$\log \frac{P_{\hat{\theta}}(s_{t+k}|S_{t+k-1}, a_{t+k-1}) P_{\hat{\theta}}(S_{t+k-1}|S_{t+k-2}, a_{t+k-2}) \ldots P_{\hat{\theta}}(S_{t+1}|s_t, a_t)}{Q_{t,k}(S_{t+1:t+k-1})}. \qquad (8)$$

Here, $Q_{t,k}$ is a probability distribution, and the inequality follows from Jensen's inequality and the concavity of $\log(\cdot)$. As in [7], the EM algorithm optimizes Eqn. (8) by alternately optimizing with respect to the distributions $Q_{t,k}$ (E-step), and the transition probabilities $P_{\hat{\theta}}(\cdot|\cdot, \cdot)$ (M-step). Optimizing with respect to the $Q_{t,k}$ variables (E-step) is achieved by setting $Q_{t,k}(S_{t+1:t+k-1}) = $

$$P_{\hat{\theta}}(S_{t+1}, \ldots, S_{t+k-1}|S_t = s_t, S_{t+k} = s_{t+k}, A_{t:t+k-1} = a_{t:t+k-1}). \qquad (9)$$

Optimizing with respect to the transition probabilities $P_{\hat{\theta}}(\cdot|\cdot, \cdot)$ (M-step) for $Q_{t,k}$ fixed as in Eqn. (9) is done by updating $\hat{\theta}$ to $\hat{\theta}_{\text{new}}$ such that $\forall\, i, j \in S, \forall\, a \in A$ we have that $P_{\hat{\theta}_{\text{new}}}(j|i, a) = stats(j, i, a) / \sum_{k \in S} stats(k, i, a)$, where $stats(j, i, a) = \sum_{t=0}^{T-1} \sum_{k=1}^{T-t} \sum_{l=0}^{k-1} \gamma^k P_{\hat{\theta}}(S_{t+l+1} = j, S_{t+l} = i|S_t = s_t, S_{t+k} = s_{t+k}, A_{t:t+k-1} = a_{t:t+k-1}) \mathbf{1}\{a_{t+l} = a\}$. Note that only the pairwise marginals $P_{\hat{\theta}}(S_{t+l+1}, S_{t+l}|S_t, S_{t+k}, A_{t:t+k-1})$ are needed in the M-step, and so it is sufficient to compute only these when optimizing with respect to the $Q_{t,k}$ variables in the E-step.